# Statistical Consistency of Ranking Methods in A Rank-Differentiable Probability Space

**Yanyan Lan**
Institute of Computing Technology
Chinese Academy of Sciences
lanyanyan@ict.ac.cn

**Jiafeng Guo**
Institute of Computing Technology
Chinese Academy of Sciences
guojiafeng@ict.ac.cn

**Xueqi Cheng**
Institute of Computing Technology
Chinese Academy of Sciences
cxq@ict.ac.cn

**Tie-Yan Liu**
Microsoft Research Asia
Tie-Yan.Liu@microsoft.com

## Abstract

This paper is concerned with the statistical consistency of ranking methods. Recently, it was proven that many commonly used pairwise ranking methods are inconsistent with the weighted pairwise disagreement loss (WPDL), which can be viewed as the true loss of ranking, even in a low-noise setting. This result is interesting but also surprising, given that the pairwise ranking methods have been shown very effective in practice. In this paper, we argue that the aforementioned result might not be conclusive, depending on what kind of assumptions are used. We give a new assumption that the labels of objects to rank lie in a rank-differentiable probability space (RDPS), and prove that the pairwise ranking methods become consistent with WPDL under this assumption. What is especially inspiring is that RDPS is actually not stronger than but similar to the low-noise setting. Our studies provide theoretical justifications of some empirical findings on pairwise ranking methods that are unexplained before, which bridge the gap between theory and applications.

## 1 Introduction

Ranking is a central problem in many applications, such as document retrieval, meta search, and collaborative filtering. In recent years, machine learning technologies called 'learning to rank' have been successfully applied. A learning-to-rank process can be described as follows. In training, a number of sets (queries) of objects (documents) are given and within each set the objects are labeled by assessors, mainly based on multi-level ratings. The target of learning is to create a model that provides a ranking over the objects that best respects the observed labels. In testing, given a new set of objects, the trained model is applied to generate a ranked list of the objects.

Ideally, the learning process should be guided by minimizing a *true loss* such as the weighted pairwise disagreement loss (WPDL) [11], which encodes people's knowledge on ranking evaluation. However, the minimization can be very difficult due to the nonconvexity of the true loss. Alternatively, many learning-to-rank methods minimize *surrogate loss functions*. For example, RankSVM [14], RankBoost [12], and RankNet [3] minimize the hinge loss, the exponential loss, and the cross-entropy loss, respectively.

In machine learning, statistical consistency is regarded as a desired property of a learning method [1, 21, 20], which reveals the statistical connection between a surrogate loss function and the true loss. Statistical consistency in the context of ranking have been actively studied in recent years

[8, 9, 19, 11, 2, 18]. According to the studies in [11], many existing pairwise ranking methods are, surprisingly, inconsistent with WPDL, even in a low-noise setting. However, as we know, the pairwise ranking methods have been shown to work very well in practice, and have been regarded as state-of-the-art even today [15, 16, 17]. For example, the experimental results in [2] show that a weighted preorder loss in RankSVM [4] can outperform a consistent surrogate loss in terms of NDCG (See Table 2 in [2]).

The contradiction between theory and application inspires us to revisit the statistical consistency of pairwise ranking methods. In particular, we will study whether there exists a new assumption on the probability space that can make statistical consistency naturally hold, and how this new assumption compares with the low-noise setting used in [11].

To perform our study, we first derive a sufficient condition for statistical consistency of ranking methods called *rank-consistency*, which is in nature very similar to *edge-consistency* in [11] and *order-preserving* in [2]. Then we give an assumption on the probability space where ratings (labels) of objects come from, which we call a rank-differentiable probability space (RDPS). Intuitively, RDPS reveals the reason why an object (denoted as object A) should be ranked higher than another object (denoted as object B). That is, the probability of any ratings consistent with the preference[1] is larger than that of its *dual ratings* (obtained by exchanging the labels of object A and object B while keeping others unchanged). We then prove that with the RDPS assumption, the weighted pairwise surrogate loss, which is a generalization of many surrogate loss functions used in existing pairwise ranking methods (e.g., the preorder loss in RankSVM [2], the exponential loss in RankBoost [12], and the logistic loss in RankNet [3]), is statistically consistent with WPDL.

Please note that our theoretical result contradicts the result obtained in [11], mainly due to the different assumptions used. What is interesting, and to some extent inspiring, is that our RDPS assumption is not stronger than the low-noise setting used in [11], and in some sense they are very similar to each other (although they focus on different aspects of the probability space). We then conducted detailed comparisons between them to gain more insights on what affects the consistency of ranking.

According to our theoretical analysis, we argue that it is not yet appropriate to draw any conclusion about the inconsistency of pairwise ranking methods, especially because it is hard to know what the probability space really is. In this sense, we think the pairwise ranking methods are still good choices for real ranking applications, due to their good empirical performances.

The rest of this paper is organized as follows. Sections 2 defines the consistency problem formally and provides a sufficient condition under which consistency with WPDL is achieved for ranking methods. Section 3 gives the main theoretical results, including formal definition of RDPS and conditions of statistical consistency of pairwise ranking methods. Further discussions on whether RDPS is a strong assumption and why our results contradict with that in [11] are presented in Section 4. Conclusions are presented in Section 5.

## 2 Preliminaries of Statistical Consistency

Let $\mathbf{x} = \{x_1, \cdots, x_m\}$ be a set of objects to be ranked. Suppose the labels of the objects are given as multi-level ratings $\mathbf{r} = (r_1, \cdots, r_m)$ from space $\mathcal{R}$, where $r_i$ denotes the label of $x_i$. Without loss of generality, we adopt K-level ratings used in [7], that is, $r_i \in \{0, 1, \cdots, K-1\}$. If $r_i > r_j$, $x_i$ should be ranked higher than $x_j$. Assume that $(\mathbf{x}, \mathbf{r})$ is a random variable of space $\mathcal{X} \times \mathcal{R}$ according to a probability measure $P$. Following existing literature, let $f$ be a ranking function that gives a score to each object to produce a ranked list and denote $\mathcal{F}$ as the space of all ranking functions.

In this paper, we adopt the *weighted pairwise disagreement loss (WPDL)* defined in [11, 10] as the true loss to evaluate $f$:

$$l_0(\alpha, G) = \sum_{i<j} a_{ij}^G 1_{\{\alpha_i \leq \alpha_j\}} + \sum_{i>j} a_{ij}^G 1_{\{\alpha_i < \alpha_j\}}, \tag{1}$$

where $\alpha = (\alpha_1, \cdots, \alpha_m) = (f(x_1), \cdots, f(x_m))$, $G$ is a directed acyclic graph (DAG for short) with edge $i \rightarrow j$ to represent the preference that $x_i$ should be ranked higher than $x_j$, and $a_{ij}^G$ is a non-negative penalty indexed by $i \rightarrow j$ on graph $G$.

Specifically, in the setting of multi-level ratings, $i \to j$ is constructed between pair $(i,j)$ with $r_i > r_j$, and $a_{ij}^G$ is thus just relevant to the labels of the two objects. For ease of representation[2], we replace $a_{ij}^G$ with $D(r_i, r_j)$, and WPDL becomes the following form:

$$l_0(f; \mathbf{x}, \mathbf{r}) = \sum_{i,j:r_i > r_j} D(r_i, r_j) 1_{\{f(x_i) - f(x_j) \le 0\}}, \tag{2}$$

where $1_{\{\cdot\}}$ is an indicator function[3] and $D(r_i, r_j)$ is a weight function s.t. (1) $\forall r_i \ne r_j, D(r_i, r_j) > 0$; (2) $\forall r_i, r_j, D(r_i, r_j) = D(r_j, r_i)$; (3) $\forall r_i < r_j < r_k, D(r_i, r_j) \le D(r_i, r_k), D(r_j, r_k) \le D(r_i, r_k)$.

The *conditional expected true risk* and the *expected true risk* of $f$ are then defined as:

$$R_0(f|\mathbf{x}) = E_{\mathbf{r}|\mathbf{x}} l_0(f; \mathbf{x}, \mathbf{r}) = \sum_{\mathbf{r} \in \mathcal{R}} l_0(f; \mathbf{x}, \mathbf{r}) P(\mathbf{r}|\mathbf{x}), \quad R_0(f) = E_{\mathbf{x}} [E_{\mathbf{r}|\mathbf{x}} l_0(f; \mathbf{x}, \mathbf{r})]. \tag{3}$$

Due to the nonconvexity of the true loss, it is infeasible to minimize the true risk in Eq.(3). As is done in the literature of machine learning, we adopt a surrogate loss $l_\Phi$ to minimize in place of $l_0$. The *conditional expected surrogate risk* and the *expected surrogate risk* of $f$ are then defined as:

$$R_\Phi(f|\mathbf{x}) = E_{\mathbf{r}|\mathbf{x}} l_\Phi(f; \mathbf{x}, \mathbf{r}) = \sum_{\mathbf{r} \in \mathcal{R}} l_\Phi(f; \mathbf{x}, \mathbf{r}) P(\mathbf{r}|\mathbf{x}), \quad R_\Phi(f) = E_{\mathbf{x}} [E_{\mathbf{r}|\mathbf{x}} l_\Phi(f; \mathbf{x}, \mathbf{r})]. \tag{4}$$

Statistical consistency is a desired property for a good surrogate loss, which measures whether the expected true risk of the ranking function obtained by minimizing a surrogate loss converges to the expected true risk of the optimal ranking in the large sample limit.

**Definition 1.** *We say a ranking method that minimizes a surrogate loss $l_\Phi$ is statistically consistent with respect to the true loss $l_0$, if $\forall \epsilon_1 > 0, \exists \epsilon_2 > 0$, such that for any ranking function $f \in \mathcal{F}$, $R_\Phi(f) \le \inf_{h \in \mathcal{F}} R_\Phi(h) + \epsilon_2$ implies $R_0(f) \le \inf_{h \in \mathcal{F}} R_0(h) + \epsilon_1$.*

We then introduce a property of the surrogate loss, called *rank-consistency*, which is a sufficient condition for the statistical consistency of the surrogate loss, as indicated by Theorem 1.

**Definition 2.** *We say a surrogate loss $l_\Phi$ is rank-consistent with the true loss $l_0$, if $\forall \mathbf{x}$, for any ranking function $f \in \mathcal{F}$ such that $R_0(f|\mathbf{x}) > \inf_{h \in \mathcal{F}} R_0(h|\mathbf{x})$, the following inequality holds:*

$$\inf_{h \in \mathcal{F}} R_\Phi(h|\mathbf{x}) < \inf\{R_\Phi(g|\mathbf{x}) : g \in \mathcal{F}, g(x_i) \le g(x_j), \text{ for } (i,j) \text{ where } f(x_i) \le f(x_j).\}. \tag{5}$$

Rank-consistency can be viewed as a generalization of *infinite sample consistency* for classification proposed in [20] (also referred to as 'classification-calibrated' in [1]) to ranking on a set of objects. It is also similar to *edge-consistent* in [11] and *order-preserving* in [2].

**Theorem 1.** *If a surrogate loss $l_\Phi$ is rank-consistent with the true loss $l_0$ on the function space $\mathcal{F}$, then it is statistically consistent with the true loss $l_0$ on $\mathcal{F}$.*

We omit the proof since it is a straightforward extension of the proof for Theorem 3 in [20]. The proof is also similar to Lemma 3, 4, 5 and Theorem 6 in [11].

# 3 Main Results

In this section, we present our main theoretical results: with a new assumption on the probability space, many commonly used pairwise ranking algorithms can be proved consistent with WPDL.

## 3.1 A Rank-Differentiable Probability Space

First, we give a new assumption named a rank-differentiable probability space (RDPS for short), with which many pairwise ranking methods will be rank-consistent with WPDL. Hereafter, we will also refer to data from RDPS as having a rank-differentiable property.

Before introducing the definition of RDPS, we give two definitions, *an equivalence class of ratings* and *dual ratings*. Intuitively, we say two ratings are equivalent if they induce the same ranking or preference relationships. And we say two ratings are the dual ratings with respect to a pair of objects, if the two ratings just exchange the ratings of the two objects while keeping the ratings of other objects unchanged. The formal definitions are given as follows.

**Definition 3.** *A ratings $\mathbf{r}$ is called equivalent to $\tilde{\mathbf{r}}$, denoted as $\mathbf{r} \sim \tilde{\mathbf{r}}$, if $\mathcal{P}(\mathbf{r}) = \mathcal{P}(\tilde{\mathbf{r}})$. Where $\mathcal{P}(\mathbf{r}) = \{(i,j) : r_i > r_j.\}$ and $\mathcal{P}(\tilde{\mathbf{r}}) = \{(i,j) : \tilde{r}_i > \tilde{r}_j.\}$ stand for the preference relationships induced by $\mathbf{r}$ and $\tilde{\mathbf{r}}$, respectively. Therefore, an equivalence class of the ratings $\mathbf{r}$, denoted as $[\mathbf{r}]$, is defined as the set of ratings which are equivalent to $\mathbf{r}$. That is, $[\mathbf{r}] = \{\tilde{\mathbf{r}} \in \mathcal{R} : \tilde{\mathbf{r}} \sim \mathbf{r}.\}$.*

**Definition 4.** *Let $R(i,j) = \{\mathbf{r} \in \mathcal{R} : r_i > r_j.\}$, $\mathbf{r}'$ is called the dual ratings of $\mathbf{r} \in R(i,j)$ with respect to $(i,j)$ if $r'_j = r_i, r'_i = r_j, r'_k = r_k, \forall k \neq i, j$.*

Now we give the definition of RDPS. An intuitive explanation on this definition is that there exists a unique equivalence class of ratings that for each induced pairwise preference relationship, the probability will be able to separate the two dual ratings with respect to that pair.

**Definition 5.** *Let $R(i,j) = \{\mathbf{r} \in \mathcal{R} : r_i > r_j.\}$, a probability space is called rank-differentiable with $(i,j)$, if for any $\mathbf{r} \in R(i,j), P(\mathbf{r}|\mathbf{x}) \geq P(\mathbf{r}'|\mathbf{x})$, and there exists at least one ratings $\mathbf{r} \in R(i,j), s.t. P(\mathbf{r}|\mathbf{x}) > P(\mathbf{r}'|\mathbf{x})$, where $\mathbf{r}'$ is the dual ratings of $\mathbf{r}$.*

**Definition 6.** *A probability space is called rank-differentiable, if there exists an equivalence class $[\mathbf{r}^*]$, s.t. $\mathcal{P}(\mathbf{r}^*) = \{(i,j) : \text{the probability space is rank-differentiable with}(i,j).\}$, where $\mathcal{P}(\mathbf{r}^*) = \{(i,j) : r_i^* > r_j^*.\}$. We will also call this probability space a RDPS or rank-differentiable with $[\mathbf{r}^*]$.*

Please note that $[\mathbf{r}^*]$ in Definition 6 is unique, which can be directly proved by Definition 3.

Definition 5 implies that if a probability space is rank-differentiable with $(i,j)$, the optimal ranking function will rank $x_i$ higher than $x_j$, as shown in the following theorem. The proof is similar to that of Theorem 4, thus we omit it here for space limitation. Hereafter, we will call this property 'separability on pairs'.

**Theorem 2.** *$\forall \mathbf{x} \in \mathcal{X}$, let $f^* \in \mathcal{F}$ be an optimal ranking function that $R_0(f^*|\mathbf{x}) = \inf_{f \in \mathcal{F}} R_0(f|\mathbf{x})$. If the probability space is rank-differentiable with $(i,j)$, we have $f^*(x_i) > f^*(x_j)$.*

Further considering the 'transitivity[4] over pairs' of a ranking function, Definition 6 implies that if a probability space is rank-differentiable with $[\mathbf{r}^*]$, the optimal ranking function will induce the same preference relationships, as shown in the following theorem.

**Theorem 3.** *$\forall \mathbf{x} \in \mathcal{X}$, let $f^* \in \mathcal{F}$ be an optimal ranking function that $R_0(f^*|\mathbf{x}) = \inf_{f \in \mathcal{F}} R_0(f|\mathbf{x})$. If the probability space is rank-differentiable with $[\mathbf{r}^*]$, for any $(i,j) \in \mathcal{P}(\mathbf{r}^*)$, we have $f^*(x_i) > f^*(x_j)$, where $\mathcal{P}(\mathbf{r}^*) = \{(i,j) : r_i^* > r_j^*.\}$.*

### 3.2 Conditions of Statistical Consistency

With RDPS as the new assumption, we study the statistical consistency of pairwise ranking methods. First, we define the weighted pairwise surrogate loss as

$$l_\Phi(f; \mathbf{x}, \mathbf{r}) = \sum_{i,j:r_i > r_j} D(r_i, r_j)\phi(f(x_i) - f(x_j)), \tag{6}$$

where $\phi$ is a convex function. The surrogate losses used in many existing pairwise ranking methods can be regarded as special cases of this weighted pairwise surrogate loss, such as the hinge loss in RankSVM [14], the exponential loss in RankBoost [12], the cross-entropy loss in RankNet [3] and the preorder loss in [2]. For the weighted pairwise surrogate loss, we get its sufficient condition of statistical consistency as shown in Theorem 5. In order to prove this theorem, we first prove Theorem 4.

**Theorem 4.** *We assume the probability space is rank-differentiable with an equivalence class $[\mathbf{r}^*]$. Suppose that $\phi(\cdot) : \mathbb{R} \rightarrow \mathbb{R}$ in the weighted pairwise surrogate loss is a non-increasing function such that $\phi(z) < \phi(-z), \forall z > 0$. $\forall \mathbf{x} \in \mathcal{X}$, let $f \in \mathcal{F}$ be a ranking function such that $R_\Phi(f|\mathbf{x}) =$*

$\inf_{h \in \mathcal{F}} R_\Phi(h|\mathbf{x})$, *then for any object pair* $(x_i, x_j), r_i^* > r_j^*$, *we have* $f(x_i) \geq f(x_j)$. *Moreover, if* $\phi(\cdot)$ *is differentiable and* $\phi'(0) < 0$, *we have* $f(x_i) > f(x_j)$.

*Proof.* (1) We assume that $f(x_i) < f(x_j)$, and define $f'$ as the function such that $f'(x_i) = f(x_j), f'(x_j) = f(x_i), f'(x_k) = f(x_k), \forall k \neq i, j$. We can then get the following equation,

$$R_\Phi(f'|\mathbf{x}) - R_\Phi(f|\mathbf{x})$$

$$= \sum_{\substack{\mathbf{r},\mathbf{r}', k:r_j < r_i < r_k \\ \mathbf{r} \in R(i,j)}} [D(r_k, r_j) - D(r_k, r_i)][\phi(f(x_k) - f(x_i)) - \phi(f(x_k) - f(x_j))][P(\mathbf{r}|\mathbf{x}) - P(\mathbf{r}'|\mathbf{x})]$$

$$+ \sum_{\substack{\mathbf{r},\mathbf{r}', k:r_j < r_k < r_i \\ \mathbf{r} \in R(i,j)}} D(r_i, r_k)[\phi(f(x_j) - f(x_k)) - \phi(f(x_i) - f(x_k))][P(\mathbf{r}|\mathbf{x}) - P(\mathbf{r}'|\mathbf{x})]$$

$$+ \sum_{\substack{\mathbf{r},\mathbf{r}', k:r_j < r_k < r_i \\ \mathbf{r} \in R(i,j)}} D(r_k, r_j)[\phi(f(x_k) - f(x_i)) - \phi(f(x_k) - f(x_j))][P(\mathbf{r}|\mathbf{x}) - P(\mathbf{r}'|\mathbf{x})]$$

$$+ \sum_{\substack{\mathbf{r},\mathbf{r}', k:r_k < r_j < r_i \\ \mathbf{r} \in R(i,j)}} [D(r_i, r_k) - D(r_j, r_k)][\phi(f(x_j) - f(x_k)) - \phi(f(x_i) - f(x_k))][P(\mathbf{r}|\mathbf{x}) - P(\mathbf{r}'|\mathbf{x})]$$

$$+ [\phi(f(x_j) - f(x_i)) - \phi(f(x_i) - f(x_j))] \sum_{\substack{\mathbf{r},\mathbf{r}', \\ \mathbf{r} \in R(i,j)}} D(r_i, r_j)[P(\mathbf{r}|\mathbf{x}) - P(\mathbf{r}'|\mathbf{x})]$$

According to the conditions of RDPS, the requirements of the weight function $D$ in Section 2 and the assumption that $\phi$ is a non-increasing function such that $\phi(z) < \phi(-z), \forall z > 0$, we can obtain

$$R_\Phi(f'|\mathbf{x}) - R_\Phi(f|\mathbf{x}) \leq [\phi(f(x_j) - f(x_i)) - \phi(f(x_i) - f(x_j))] \sum_{\substack{\mathbf{r},\mathbf{r}', \\ \mathbf{r} \in R(i,j)}} D(r_i, r_j)[P(\mathbf{r}|\mathbf{x}) - P(\mathbf{r}'|\mathbf{x})] < 0.$$

This is a contradiction with $R_\Phi(f) = \inf_{h \in \mathcal{F}} R_\Phi(h|\mathbf{x})$. Therefore, we have proven that $f(x_i) \geq f(x_j)$.

(2) Now we assume that $f(x_i) = f(x_j) = f_0$. From the assumption $R_\Phi(f|\mathbf{x}) = \inf_{h \in \mathcal{F}} R_\Phi(h|\mathbf{x})$, we can get $\left.\frac{\partial R_\Phi(f|\mathbf{x})}{\partial f(x_i)}\right|_{f_0} = 0, \left.\frac{\partial R_\Phi(f|\mathbf{x})}{\partial f(x_j)}\right|_{f_0} = 0$. Accordingly, we can obtain the two following equations:

$$\sum_{\substack{\mathbf{r},\mathbf{r}', \\ \mathbf{r} \in R(i,j)}} A_1 P(\mathbf{r}|\mathbf{x}) + A_2 P(\mathbf{r}'|\mathbf{x}) = 0, \qquad \sum_{\substack{\mathbf{r},\mathbf{r}', \\ \mathbf{r} \in R(i,j)}} B_1 P(\mathbf{r}|\mathbf{x}) + B_2 P(\mathbf{r}'|\mathbf{x}) = 0, \qquad (7)$$

where,

$$A_1 = B_2 = \sum_{k:r_j < r_i < r_k} D(r_k, r_i)[-\phi'(f(x_k) - f_0)] + \sum_{k:r_j < r_k < r_i} D(r_i, r_k)\phi'(f_0 - f(x_k))$$

$$+ \sum_{k:r_k < r_j < r_i} D(r_i, r_k)\phi'(f_0 - f(x_k)) + D(r_i, r_j)\phi'(0).$$

$$A_2 = B_1 = \sum_{k:r_j < r_i < r_k} D(r_k, r_j)[-\phi'(f(x_k) - f_0)] + \sum_{k:r_j < r_k < r_i} D(r_k, r_j)[-\phi'(f(x_k) - f_0)]$$

$$+ \sum_{k:r_k < r_j < r_i} D(r_j, r_k)\phi'(f_0 - f(x_k)) + D(r_i, r_j)[-\phi'(0)].$$

If $\phi'(0) < 0$, based on the requirements of RDPS and the weight function $D$, we can get

$$\sum_{\substack{\mathbf{r},\mathbf{r}', \\ \mathbf{r} \in R(i,j)}} (A_1 - B_1)P(\mathbf{r}|\mathbf{x}) + (A_2 - B_2)P(\mathbf{r}'|\mathbf{x})$$

$$= \sum_{\substack{\mathbf{r},\mathbf{r}', \\ \mathbf{r} \in R(i,j)}} (A_1 - A_2)[P(\mathbf{r}|\mathbf{x}) - P(\mathbf{r}'|\mathbf{x})] \leq 2\phi'(0) \sum_{\substack{\mathbf{r},\mathbf{r}', \\ \mathbf{r} \in R(i,j)}} D(r_i, r_j)[P(\mathbf{r}|\mathbf{x}) - P(\mathbf{r}'|\mathbf{x})] < 0.$$

This is a contradiction with Eq.(7). Therefore, we actually have proven that $f(x_i) > f(x_j)$. $\square$

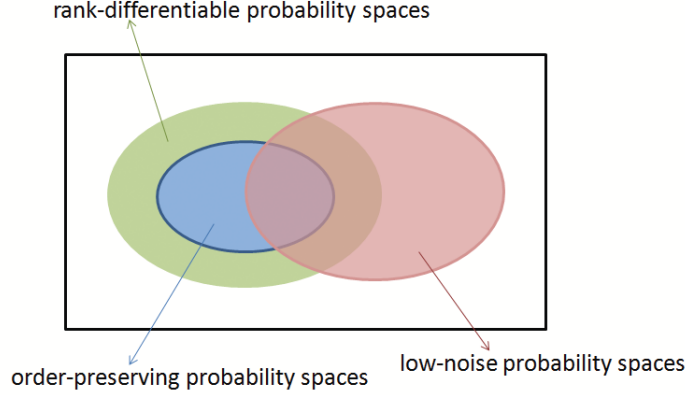

Figure 1: Relationships among order-preserving, rank-differentiable and low-noise.

**Theorem 5.** *Let $\phi(\cdot)$ be a non-negative, non-increasing and differentiable function such that $\phi'(0) < 0$. Then the weighted pairwise surrogate loss is consistent with WPDL under the assumption of RDPS.*

*Proof.* We assume that the probability space is rank-differentiable with an equivalence class $[\mathbf{r}^*]$. Then for any object pair $(x_i, x_j), r_i^* > r_j^*$, we are going to prove that

$$R_{\Phi|\mathbf{x}}^* = \inf_{h \in \mathcal{F}} R_\Phi(h|\mathbf{x}) < \inf\{R_\Phi(f|\mathbf{x}) : f \in \mathcal{F}, f(x_i) \le f(x_j).\} \tag{8}$$

because from Theorem 3 this implies the rank-consistency condition in Eq.(5) holds.

Suppose Eq.(8) is not true, then we can find a sequence of functions $\{f_m\}$ such that $0 = f_m(x_i) \le f_m(x_j)$, and $\lim_m R_\Phi(f_m|\mathbf{x}) = R_{\Phi|\mathbf{x}}^*$. We can further select a subsequence such that for each pair $(i, j)$, $f_m(x_i) - f_m(x_j)$ converges (may also converge to $\pm\infty$). This leads to a limit function $f$, with properly defined $f(x_i) - f(x_j)$, even when either $f(x_i)$ or $f(x_j)$ is $\pm\infty$. This implies that $R_\Phi(f|\mathbf{x}) = R_{\Phi|\mathbf{x}}^*$ and $0 = f(x_i) \le f(x_j)$. However, this violates Theorem 4. Thus, Eq.(8) is true. Therefore, we have proven that the weighted pairwise surrogate loss is consistent with WPDL. $\square$

Many commonly used pairwise surrogate losses, such as the preorder loss in RankSVM [2], the exponential loss in RankBoost [12], and the logistic loss in RankNet[3], satisfy the conditions in Theorem 5, thus they are consistent with WPDL. In other words, we have shown that statistical consistency of pairwise ranking methods is achieved under the assumption of RDPS.

## 4 Discussions

In Section 3, we have shown that statistical consistency of pairwise ranking methods is achieved with the assumption of RDPS. Considering the contradicting conclusion drawn in [11], a natural question is whether the RDPS is stronger than the low-noise setting used in [11]. In this section we will make some discussions on this issue.

### 4.1 Relationships of RDPS with Previous Work

Here, we discuss the relationships between the rank-differentiable property and the assumptions used in some previous works (including the order-preserving property in [19] and the low-noise setting in [11]). According to our analysis, we find that the rank-differentiable property is not a strong assumption on the probability space. Actually, it is a weaker assumption than the order-preserving property and is very similar to the low-noise setting. A sketch map of the relationships between the three assumptions is presented in Figure 1, where the low-noise probability spaces stands for a set of spaces satisfying the low-noise setting. Detailed discussions are given as follows.

1. **Rank-Differentiable vs. Order-Preserving**

The rank-differentiable property is defined on the space of multi-level ratings while the order-preserving property is defined on the permutation space. To understand their relationship, we need to put them onto the same space. Actually, we can restrict the space of multi-level ratings to the permutation space by setting $K = m - 1$ and requiring the ratings of each two objects to be different. After doing so, it is not difficult to see that the rank-differentiable property is weaker than the order-preserving property, as shown in the following theorem.

**Theorem 6.** *Let $K = m - 1$. For each permutation $y \in \mathcal{Y}$, where $y(i)$ stands for the position of $x_i$, define the corresponding ratings $\mathbf{r}^y = (r_1^y, \cdots, r_m^y)$ as $r_i^y = m - y(i), i = 1, \cdots, n$. Assume that $P(\mathbf{r}^y) = P(y)$, and $P(\mathbf{r}) = 0$ if there does not exist a permutation $y$ s.t. $\mathbf{r} = \mathbf{r}^y$. If the probability space is order-preserving with respect to $m-1$ pairs $(j_1, j_2), (j_2, j_3), \cdots, (j_{m-1}, j_m)$, it is rank-differentiable with the equivalence class $[\mathbf{r}^*]$, where $r_{j_i}^* > r_{j_{i+1}}^*, i = 1, \cdots, m$, but the converse is not always true.*

2. **Rank-Differentiable vs. Low-Noise**

The rank-differentiable property is defined on the space of multi-level ratings while the low-noise setting is defined on the space of DAGs. According to the correspondence between ratings and DAGs (as stated in Section 2), we can restrict the space of DAGs to the space of multi-level ratings. Consequently, we obtain the relationship between the rank-differentiable property and the low-noise setting as follows:

(1) Mathematically, the inequalities in the low-noise setting can be viewed as the combinations of the corresponding inequalities in the rank-differentiable property. They are similar to each other in their forms and the rank-differentiable property is not stronger than the low-noise setting.

(2) Intuitively, the rank-differentiable property induces 'separability on pairs' and 'transitivity over pairs' as described in Theorem 2 and 3, while the low-noise setting aims to explicitly express the transitivity over pairs, but fails in achieving it.

Let us use an example to illustrate the above points. Suppose there are three objects to be ranked in the setting of three-level ratings ($K = 3$). Furthermore, suppose that the ratings of every two objects are different and all the graphs are fully connected DAGs in the setting of [11]. We order the ratings and DAGs as:

$$\mathbf{r}_1 = (2, 1, 0), \mathbf{r}_2 = (1, 2, 0), \mathbf{r}_3 = (2, 0, 1), \mathbf{r}_4 = (0, 2, 1), \mathbf{r}_5 = (1, 0, 2), \mathbf{r}_6 = (0, 1, 2).$$
$$G_1 = \{(1\to2), (2\to3), (1\to3)\}, G_2 = \{(2\to1), (1\to3), (2\to3)\}, G_3 = \{(1\to3), (3\to2), (1\to2)\},$$
$$G_4 = \{(2\to3), (3\to1), (2\to1)\}, G_5 = \{(3\to1), (1\to2), (3\to2)\}, G_6 = \{(3\to2), (2\to1), (3\to1)\},$$

Therefore $\mathbf{r}_i, G_i$ have one-to-one correspondence, we can set the probability as $P(\mathbf{r}_i|\mathbf{x}) = P(G_i|\mathbf{x}) = P_i$ and define $a_{kl}^{G_i} = D(r_{ik}, r_{il}), i = 1, \cdots, 6; k, l = 1, 2, 3$.

Considering conditions in the definition of RDPS, rank-differentiable with $[r_1]$ requires the following inequalities to hold and at least one inequalities in (9) and (10) to hold strictly.

$$P_1 - P_2 \geq 0, P_3 - P_4 \geq 0, P_5 - P_6 \geq 0, \tag{9}$$

$$P_4 - P_6 \geq 0, P_2 - P_5 \geq 0, P_1 - P_3 \geq 0, \tag{10}$$

We assume there are edges $1 \to 2$ and $2 \to 3$ in the difference graph. Then the low-noise setting in Definition 8 of [11] requires that $a_{13} - a_{31} \geq a_{12} - a_{21} + a_{23} - a_{32}$, where,

$$a_{12} - a_{21} = D(2, 1)(P_1 - P_2) + D(2, 0)(P_3 - P_4) + D(1, 0)(P_5 - P_6),$$
$$a_{23} - a_{32} = D(2, 1)(P_4 - P_6) + D(2, 0)(P_2 - P_5) + D(1, 0)(P_1 - P_3),$$
$$a_{13} - a_{31} = D(2, 1)(P_3 - P_5) + D(2, 0)(P_1 - P_6) + D(1, 0)(P_2 - P_4).$$

According to the above example,

(1) $a_{12} - a_{21}$ and $a_{23} - a_{32}$ are exactly the combinations of the terms in (9) and (10), respectively. Thus, if the probability space is rank-differentiable with $[r_1]$, we can only get $a_{12} - a_{21} > 0, a_{23} - a_{32} > 0$, but not the inequalities in the low-noise setting. This indicates that our rank-differentiable property is not stronger than the low-noise setting.

(2) With the assumption that $a_{ij} - a_{ji} > 0$ can guarantee the optimal ranking with which $x_i$ is ranked higher than $x_j$, it seems that the low-noise setting intends to make the preferences of $1 \rightarrow 2$ and $2 \rightarrow 3$ transitive to $1 \rightarrow 3$. However, the assumption is not always true. Instead, the rank-differentiable property can naturally induce the 'transitivity over pairs' (See Theorem 2 and 3). In this sense, the rank-differentiable property is much more powerful than the low-noise setting, although not stronger.

### 4.2 Explanation on Theoretical Contradiction

On one hand, different conclusions on the consistency of pairwise ranking methods have been obtained in our work and in [11]. On the other hand, we have shown that there exists an connection between the rank-differentiable property and the low-noise setting (see Figure 1). Therefore, one may get confused by the contradicting results and may wonder what will happen if a probability space satisfies both the rank-differentiable property and the low-noise setting. In this subsection, we will make discussions on this issue.

Please note that we adopt the multi-level ratings as the labeling strategy (as stated clearly in Section 2) in our analysis. With this setting, the graph space $\mathcal{G}$ in [11] will not contain all the DAGs. For example, considering a three-graph case, the graph $G_2 = \{(1, 2, 3) : (2 \rightarrow 3), (3 \rightarrow 1)\}$ in the proof of Theorem 11 of [11] (the main negative result on the consistency of pairwise surrogate losses) actually does not exist. That is because if $2 \rightarrow 3$ and $3 \rightarrow 1$ exist in a graph $G$, we can get that $r_2 > r_3, r_3 > r_1$ according to the correspondence between graphs and ratings as stated in Section 2. Therefore, we can immediately get $r_2 > r_1$. Once again according to the correspondence between graphs and ratings, we will get that $2 \rightarrow 1$ should be contained in graph $G$, which contradicts with $G_2$. Thus, $G_2$ will not exist in the setting of multi-level ratings. However, in the proof of [11], they do not take the constraint of multi-level ratings into consideration, thus deduce contradict results.

From the above discussions, we can see that our theoretical results contradict with that in [11] mainly because the two works consider different settings and assumptions. If a probability space satisfies both the rank-differentiable property and the low-noise setting, the pairwise ranking methods will be consistent with WPDL in the setting of multi-level ratings but inconsistent in the setting of DAGs. One may argue that the setting of multi-level ratings is not as general as the DAG setting, however, please note that multi-level ratings are the dominant setting in the literature of 'learning to rank' [13, 16, 15, 6] and have been widely used in many applications such as web search and document retrieval [17, 5]. Therefore, we think the setting of multi-level ratings is general enough and our result has its value to the mainstream research of learning to rank.

To sum up, based on all the discussions in this paper, we argue that it is not yet appropriate to draw any conclusion about the inconsistency of pairwise ranking methods, especially because it is hard to know what the probability space really is. In this sense, we think the pairwise ranking methods are still good choices for real ranking applications, due to their good empirical performances.

## 5  Conclusions

In this paper, we have discussed the statistical consistency of ranking methods. Specifically, we argue that the previous results on the inconsistency of commonly-used pairwise ranking methods are not conclusive, depending on the assumptions about the probability space. We then propose a new assumption, which we call a rank-differentiable probability space (RDPS), and prove that the pairwise ranking methods are consistent with the same true loss as in previous studies under this assumption. We show that RDPS is not a stronger assumption than the assumptions used in previous work, indicating that our finding is similarly reliable to previous ones.

### Acknowledgments

This research work was funded by the National Natural Science Foundation of China under Grant No. 60933005, No. 61173008, No. 61003166 , No. 61203298 and 973 Program of China under Grants No. 2012CB316303.

## Footnotes

[1]Here, consistency with the preference means that the rating of object A is larger than that of object B.

[2]Here we do not distinguish $i > j$ and $i < j$, because they are just introduced to avoid minor technical issues as stated in [11]. Furthermore, it will not influence the consistency results.

[3]$1_A = 1$, if $A$ is true and $1_A = 0$, if $A$ is false.

[4]Transitivity means that if $x_i$ is ranked higher than $x_j$ and $x_j$ is ranked higher than $x_k$, $x_i$ must be ranked higher than $x_k$.

# References

[1] P. L. Bartlett, M. I. Jordan, and J. D. McAuliffe. Convexity, classification, and risk bounds. *Journal of the American Statistical Association*, 101(473):138–156, 2006.

[2] D. Buffoni, C. Calauzenes, P. Gallinari, and N. Usunier. Learning scoring functions with order-preserving losses and standardized supervision. In *Proceedings of the 28th International Conference on Machine Learning (ICML 2011)*, pages 825–832, 2011.

[3] C. Burges, T. Shaked, E. Renshaw, A. Lazier, M. Deeds, N. Hamilton, and G. Hullender. Learning to rank using gradient descent. In *Proceedings of the 22th International Conference on Machine Learning (ICML 2005)*, pages 89–96, 2005.

[4] O. Chapelle. Training a support vector machine in the primal. *Neural Computation*, 19:1155–1178, 2007.

[5] O. Chapelle and Y. Chang. Yahoo! learning to rank challenge overview. *Journal of Machine Learning Research - Proceedings Track*, 14:1–24, 2011.

[6] O. Chapelle, Y. Chang, and T.-Y. Liu. Future directions in learning to rank. *Journal of Machine Learning Research - Proceedings Track*, 14:91–100, 2011.

[7] W. Chen, T.-Y. Liu, Y. Lan, Z. Ma, and H. Li. Ranking measures and loss functions in learning to rank. In *24th Annual Conference on Neural Information Processing Systems (NIPS 2009)*, pages 315–323, 2009.

[8] S. Clémençon, G. Lugosi, and N. Vayatis. Ranking and scoring using empirical risk minimization. In *Proceedings of the 18th Annual Conference on Learning Theory (COLT 2005)*, volume 3559, pages 1–15, 2005.

[9] D. Cossock and T. Zhang. Subset ranking using regression. In *Proceedings of the 19th Annual Conference on Learning Theory (COLT 2006)*, pages 605–619, 2006.

[10] O. Dekel, C. D. Manning, and Y. Singer. Log-linear models for label ranking. In *18th Annual Conference on Neural Information Processing Systems (NIPS 2003)*, 2003.

[11] J. C. Duchi, L. W. Mackey, and M. I. Jordan. On the consistency of ranking algorithms. In *Proceedings of the 27th International Conference on Machine Learning (ICML 2010)*, pages 327–334, 2010.

[12] Y. Freund, R. Iyer, R. E. Schapire, and Y. Singer. An efficient boosting algorithm for combining preferences. *Journal of Machine Learning Research*, 4:933–969, 2003.

[13] R. Herbrich, K. Obermayer, and T. Graepel. Large margin rank boundaries for ordinal regression. In *Advances in Large Margin Classifiers.*, pages 115–132, 1999.

[14] T. Joachims. Optimizing search engines using clickthrough data. In *Proceedings of the 8th ACM SIGKDD International Conference on Knowledge Discovery and Data Mining (KDD 2002)*, pages 133–142, 2002.

[15] H. Li, T.-Y. Liu, and C. Zhai. Learning to rank for information retrieval (lr4ir 2008). *SIGIR Forum*, 42:76–79, 2008.

[16] T.-Y. Liu. Learning to rank for information retrieval. *Foundation and Trends on Information Retrieval*, 3:225–331, 2009.

[17] T.-Y. Liu, J. Xu, T. Qin, W.-Y. Xiong, and H. Li. Letor: Benchmark dataset for research on learning to rank for information retrieval. In *SIGIR '07 Workshop*, San Francisco, 2007. Morgan Kaufmann.

[18] P. D. Ravikumar, A. Tewari, and E. Yang. On ndcg consistency of listwise ranking methods. *Journal of Machine Learning Research - Proceedings Track*, 15:618–626, 2011.

[19] F. Xia, T.-Y. Liu, J. Wang, W. S. Zhang, and H. Li. Listwise approach to learning to rank - theory and algorithm. In *Proceedings of the 25th International Conference on Machine Learning (ICML 2008)*, 2008.

[20] T. Zhang. Statistical analysis of some multi-category large margin classification methods. *Journal of Machine Learning Research*, 5:1225–1251, 2004.

[21] T. Zhang. Statistical behavior and consistency of classification methods based on convex risk minimization. *Annuals of Statistics*, 32:56–85, 2004.

